# Reinforcement Learning for Spoken Dialogue Systems

**Satinder Singh**
AT&T Labs

**Michael Kearns**
AT&T Labs

**Diane Litman**
AT&T Labs

**Marilyn Walker**
AT&T Labs

{baveja,mkearns,diane,walker}@research.att.com

## Abstract

Recently, a number of authors have proposed treating dialogue systems as Markov decision processes (MDPs). However, the practical application of MDP algorithms to dialogue systems faces a number of severe technical challenges. We have built a general software tool (RLDS, for Reinforcement Learning for Dialogue Systems) based on the MDP framework, and have applied it to dialogue corpora gathered from two dialogue systems built at AT&T Labs. Our experiments demonstrate that RLDS holds promise as a tool for "browsing" and understanding correlations in complex, temporally dependent dialogue corpora.

## 1 Introduction

Systems in which human users speak to a computer in order to achieve a goal are called *spoken dialogue systems*. Such systems are some of the few realized examples of open-ended, real-time, goal-oriented interaction between humans and computers, and are therefore an important and exciting testbed for AI and machine learning research. Spoken dialogue systems typically integrate many components, such as a speech recognizer, a database back-end (since often the goal of the user is to retrieve information), and a dialogue strategy. In this paper we are interested in the challenging problem of automatically inferring a good dialogue strategy from dialogue corpora.

Research in dialogue strategy has been perhaps necessarily ad-hoc due to the open-ended nature of dialogue system design. For example, a common and critical design choice is between a system that always prompts the user to select an utterance from fixed menus (*system initiative*), and one that attempts to determine user intentions from unrestricted utterances (*mixed initiative*). Typically a system is built that explores a few alternative strategies, this system is tested, and conclusions are drawn regarding which of the tested strategies is best for that domain [4, 7, 2]. This is a time-consuming process, and it is difficult to rigorously compare and evaluate alternative systems in this fashion, much less design improved ones.

Recently, a number of authors have proposed treating dialogue design in the formalism of Markov decision processes (MDPs)[1, 3, 7]. In this view, the population of users defines the stochastic environment, a dialogue system's actions are its (speech-synthesized) utterances and database queries, and the state is represented by the entire dialogue so far. The goal is to design a dialogue system that takes actions so as to maximize some measure of reward. Viewed in this manner, it becomes possible, at least in principle, to apply the framework and algorithms of reinforcement learning (RL) to find a good dialogue strategy.

However, the practical application of RL algorithms to dialogue systems faces a number of severe technical challenges. First, representing the dialogue state by the entire dialogue so

far is often neither feasible nor conceptually useful, and the so-called belief state approach is not possible, since we do not even know what features are required to represent the belief state. Second, there are many different choices for the reward function, even among systems providing very similar services to users. Previous work [7] has largely dealt with these issues by imposing a priori limitations on the features used to represent approximate state, and then exploring just one of the potential reward measures.

In this paper, we further develop the MDP formalism for dialogue systems, in a way that does not solve the difficulties above (indeed, there is no simple "solution" to them), but allows us to attenuate and quantify them by permitting the investigation of different notions of approximate state and reward. Using our expanded formalism, we give one of the first applications of RL algorithms to real data collected from multiple dialogue systems. We have built a general software tool (RLDS, for Reinforcement Learning for Dialogue Systems) based on our framework, and applied it to dialogue corpora gathered from two dialogue systems built at AT&T Labs, the TOOT system for voice retrieval of train schedule information [4] and the ELVIS system for voice retrieval of electronic mail [7].

Our experiments demonstrate that RLDS holds promise not just as a tool for the end-to-end automated synthesis of complicated dialogue systems from passive corpora — a "holy grail" that we fall far short of here[1] — but more immediately, as a tool for "browsing" and understanding correlations in complex, temporally dependent dialogue corpora. Such correlations may lead to incremental but important improvements in existing systems.

## 2 The TOOT and ELVIS Spoken Dialogue Systems

The TOOT and ELVIS systems were implemented using a general-purpose platform developed at AT&T, combining a speaker-independent hidden Markov model speech recognizer, a text-to-speech synthesizer, a telephone interface, and modules for specifying data-access functions and dialogue strategies. In TOOT, the data source is the Amtrak train schedule web site, while in ELVIS, it is the electronic mail spool of the user.

In a series of controlled experiments with human users, dialogue data was collected from both systems, resulting in 146 dialogues from TOOT and 227 dialogues from ELVIS. The TOOT experiments varied strategies for information presentation, confirmation (whether and how to confirm user utterances) and initiative (system vs. mixed), while the ELVIS experiments varied strategies for information presentation, for summarizing email folders, and initiative. Each resulting dialogue consists of a series of system and user utterances augmented by observations derived from the user utterances and the internal state of the system. The system's utterances (*actions*) give requested information, ask for clarification, provide greetings or instructions, and so on. The observations derived from the user's utterance include the speech-recognizer output, the corresponding log-likelihood score, the semantic labels assigned to the recognized utterances (such as the desired train departure and arrival cities in TOOT, or whether the user prefers to hear their email ordered by date or sender in ELVIS); indications of user barge-ins on system prompts; and many more. The observations derived from the internal state include the grammar used by the speech recognizer during the turn, and the results obtained from a query to the data source. In addition, each dialogue has an associated survey completed by the user that asks a variety of questions relating to the user's experience. See [4, 7] for details.

## 3 Spoken Dialogue Systems and MDPs

Given the preceding discussion, it is natural to formally view a dialogue as a sequence $d$

$$d = (a_1, \vec{o}_1, r_1), (a_2, \vec{o}_2, r_2), \ldots, (a_t, \vec{o}_t, r_t).$$

Here $a_i$ is the action taken by the system (typically a speech-synthesized utterance, and less frequently, a database query) to start the $i$th exchange (or *turn*, as we shall call it), $\vec{o}_i$ consists of all the observations logged by the system on this turn, as discussed in the last section, and $r_i$ is the reward received on this turn. As an example, in TOOT a typical turn might indicate that the action $a_i$ was a system utterance requesting the departure city, and the $\vec{o}_i$ might indicate several observations: that the recognized utterance was "New York", that the log-likelihood of this recognition was $-2.7$, that there was another unrecognized utterance as well, and so on. We will use $d[i]$ to denote the prefix of $d$ that ends following the $i$th turn, and $d \cdot (a, \vec{o}, r)$ to denote the one-turn extension of dialogue $d$ by the turn $(a, \vec{o}, r)$. The scope of the actions $a_i$ and observations $\vec{o}_i$ is determined by the implementation of the systems (e.g. if some quantity was not logged by the system, we will not have access to it in the $\vec{o}_i$ in the data). Our experimental results will use rewards derived from the user satisfaction surveys gathered for the TOOT and ELVIS data.

We may view any dialogue $d$ as a trajectory in a well-defined *true* MDP $M$. The states [2] of $M$ are all possible dialogues, and the actions are all the possible actions available to the spoken dialogue system (utterances and database queries). Now from any state (dialogue) $d$ and action $a$, the only possible next states (dialogues) are the one-turn extensions $d \cdot (a, \vec{o}, r)$. The probability of transition from $d$ to $d \cdot (a, \vec{o}, r)$ is exactly the probability, over the stochastic ensemble of users, that $\vec{o}$ and $r$ would be generated following action $a$ in dialogue $d$.

It is in general impractical to work directly on $M$ due to the unlimited size of the state (dialogue) space. Furthermore, $M$ is not known in advance and would have to be estimated from dialogue corpora. We would thus like to permit a flexible notion of *approximate* states. We define *state estimator* SE to be a mapping from any dialogue $d$ into some space $\mathcal{S}$. For example, a simple state estimator for TOOT might represent the dialogue state with boolean variables indicating whether certain pieces of information had yet been obtained from the user (departure and arrival cities, and so on), and a continuous variable tracking the average log-likelihood of the recognized utterances so far. Then $SE(d)$ would be a vector representing these quantities for the dialogue $d$. Once we have chosen a state estimator SE, we can transform the dialogue $d$ into an $\mathcal{S}$-*trajectory*, starting from the initial empty state $s_0 \in \mathcal{S}$:

$$s_0 \to_{a_1} SE(d[1]) \to_{a_2} SE(d[2]) \to_{a_3} \cdots \to_{a_t} SE(d[t])$$

where the notation $\to_{a_i} SE(d[i])$ indicates a transition to $SE(d[i]) \in \mathcal{S}$ following action $a_i$. Given a set of dialogues $d_1, \ldots, d_n$, we can construct the *empirical* MDP $\hat{M}_{SE}$. The state space of $\hat{M}_{SE}$ is $\mathcal{S}$, the actions are the same as in $M$, and the probability of transition from $s$ to $s'$ under action $a$ is exactly the empirical probability of such a transition in the $\mathcal{S}$-trajectories obtained from $d_1, \ldots, d_n$. Note that we can build $\hat{M}_{SE}$ from dialogue corpora, solve for its optimal policy, and analyze the resulting value function.

The point is that by choosing SE carefully, we hope that the empirical MDP $\hat{M}_{SE}$ will be a good approximation of $M$. By this we mean that $\hat{M}_{SE}$ *renders dialogues (approximately) Markovian*: the probability in $M$ of transition from any dialogue $d$ to any one-turn extension $d \cdot (a, \vec{o}, r)$ is (approximately) the probability of transition from $SE(d)$ to $SE(d \cdot (a, \vec{o}, r))$ in $\hat{M}_{SE}$. We hope to find state estimators SE which render dialogues approximately Markovian, but for which the amount of data and computation required to find good policies in $\hat{M}_{SE}$ will be greatly reduced compared to working directly in dialogue space.

While conceptually appealing, this approach is subject to at least three important caveats: First, the approach is theoretically justified only to the extent that the chosen state estimator renders dialogues Markovian. In practice, we hope that the approach is robust, in that "small" violations of the Markov property will still produce useful results. Second, while

state estimators violating the Markov property may lead to meaningful insights, they cannot be directly compared. For instance, if the optimal value function derived from one state estimator is larger than the optimal value function for another state estimator, we *cannot* necessarily conclude that the first is better than the second. (This can be demonstrated formally.) Third, even with a Markovian state estimator SE, data that is sparse with respect to SE limits the conclusions we can draw; in a large space $S$, certain states may be so infrequently visited in the dialogue corpora that we can say nothing about the optimal policy or value function there.

## 4   The RLDS System

We have implemented a software tool (written in C) called RLDS that realizes the above formalism. RLDS users specify an input file of sample dialogues; the dialogues include the rewards received at each turn. Users also specify input files defining $S$ and a state estimator SE. The system has command-line options that specify the discount factor to be used, and a lower bound on the number of times a state $s \in S$ must be visited in order for it to be included in the empirical MDP $\hat{M}_{SE}$ (to control overfitting to sparse data). Given these inputs and options, RLDS converts the dialogues into $S$-trajectories, as discussed above. It then uses these trajectories to compute the empirical MDP $\hat{M}_{SE}$ specified by the data — that is, the data is used to compute next-state distributions and average reward in the obvious way. States with too few visits are pruned from $\hat{M}_{SE}$. RLDS then uses the standard value iteration algorithm to compute the optimal policy and value function [6] for $\hat{M}_{SE}$, all using the chosen discount factor.

## 5   Experimental Results

The goal of the experiments reported below is twofold: first, to confirm that our RLDS methodology and software produce intuitively sensible policies; and second, to use the value functions computed by the RLDS software to discover and understand correlations between dialogue properties and performance. We have space to present only a few of our many experiments on TOOT and ELVIS data.

Each experiment reported below involves choosing a state estimator, running RLDS using either the TOOT or ELVIS data, and then analyzing the resulting policy and value function. For the TOOT experiments, the reward function was obtained from a question in the user satisfaction survey: the last turn in a dialogue receives a reward of $+1$ if the user indicated that they would use the system again, a reward of $0$ if the user answered "maybe", and a reward of $-1$ if the user indicated that they would not use the system again. All turns other than the last receive reward $0$ (i.e., a reward is received only at the end of a dialogue). For the ELVIS experiments, we used a summed (over several questions) user-satisfaction score to reward the last turn in each dialogue (this score ranges between 8 and 40).

**Experiment 1 (A Sensible Policy):** In this initial "sanity check" experiment, we created a state estimator for TOOT whose boolean state variables track whether the system knows the value for the following five informational attributes: arrival city (denoted AC), departure city (DC), departure date (DD), departure hour (DH), and whether the hour is AM or PM (AP) [3]. Thus, if the dialogue so far includes a turn in which TOOT prompts the user for their departure city, and the speech recognizer matches the user utterance with "New York", the boolean state variable GotDC? would be assigned a value of 1. Note that this ignores the actual values of the attributes. In addition, there is another boolean variable called ConfirmedAll? that is set to 1 if and only if the system took action ConfirmAll (which prompts the user to explicitly verify the attribute values perceived by TOOT) and perceived a "yes" utterance in response. Thus, the state vector is simply the binary vector

```
[ GotAC? , GotAP? , GotDC? , GotDD? , GotDH? , ConfirmedAll? ]
```

Among the actions (the system utterances) available to TOOT are prompts to the user to specify values for these informational attributes; we shall denote these actions with labels AskDC, AskAC, AskDD, AskDH, and AskAP. The system takes several other actions that we shall mention as they arise in our results.

The result of running RLDS was the following policy, where we have indicated the action to be taken from each state:

```
[0,0,0,0,0,0]: SayGreeting [1,0,0,0,0,0]: AskDC  [1,0,1,0,0,0]: AskAP
[1,0,1,1,0,0]: AskDH       [0,0,0,1,1,0]: AskAP  [1,0,0,1,1,0]: AskAP
[0,1,0,1,1,0]: AskAll      [1,1,0,1,1,0]: AskAll [1,0,1,1,1,0]: AskAP
[1,1,1,1,1,0]: ConfirmAll  [1,1,1,1,1,1]: Close
```

Thus, RLDS finds a sensible policy, always asking the user for information which it has not already received, confirming the user's choices when it has all the necessary information, and then presenting the closest matching train schedule and closing the dialogue (action Close). Note that in some cases it chooses to ask the user for values for all the informational attributes even though it has values for some of them. It is important to emphasize that this policy was derived purely through the application of RLDS to the dialogue data, without any knowledge of the "goal" of the system. Furthermore, the TOOT data is such that the empirical MDP built by RLDS for this state estimator does include actions considerably less reasonable than those chosen above from many states. Examples include confirming the values of specific informational attributes such as DC (since we do not represent whether such confirmations were successful, this action would lead to infinite loops of confirmation), and requesting values for informational attributes for which we already have values (such actions appear in the empirical MDP due to speech recognition errors). The mere fact that RLDS was driven to a sensible policy that avoided these available pitfalls indicates a correlation between the chosen reward measure (whether the user would use the system again) and the intuitive system goal of obtaining a completely specified train trip. It is interesting to note that RLDS finds it better to confirm values for all 5 attributes when it has them, as opposed to simply closing the dialogue without confirmation.

In a similar experiment on ELVIS, RLDS again found a sensible policy that summarizes the user's inbox at the beginning of the dialogue, goes on to read the relevant e-mail messages until done, and then closes.

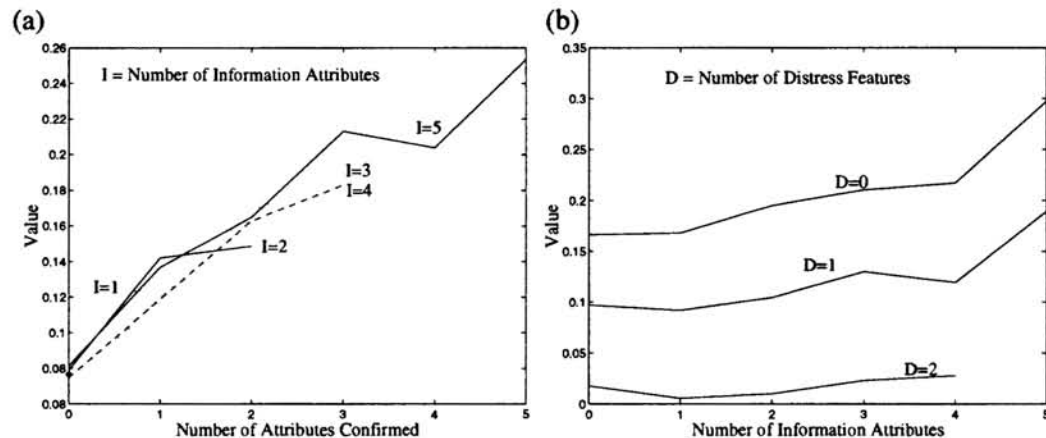

Figure 1: a) Role of Confirmation. b) Role of Distress Features (indicators that the dialogue is in trouble). See description of Experiments 2 and 3 respectively in the text for details.

**Experiment 2 (Role of Confirmation):** Here we explore the effect of confirming with the user the values that TOOT perceives for the informational attributes — that is, whether the

trade-off between the increased confidence in the utterance and the potential annoyance to the user balances out in favor of confirmation or not (for the particular reward function we are using). To do so, we created a simple state estimator with just two state variables. The first variable counts the number of the informational attributes (DC, AC, etc.) that TOOT believes it has obtained, while the second variable counts the number of these that have been confirmed with the user. Figure 1(a) presents the optimal value as a function of the number of attributes confirmed. Each curve in the plot corresponds to a different setting of the first state variable. For instance, the curve labeled with "I=3" corresponds to the states where the system has obtained 3 informational attributes. We can make two interesting observations from this figure. First, the value function grows roughly linearly with the number of confirmed attributes. Second, and perhaps more startlingly, the value function has only a weak dependence on the first feature — the value for states when some number of attributes have been *confirmed* seems independent of how many attributes (the system believes) have been *obtained*. This is evident from the lack of separation between the plots for varying values of the state variable I. In other words, our simple (and preliminary) analysis suggests that for our reward measure, confirmed information influences the value function much more strongly than unconfirmed information. We also repeated this experiment replacing attribute confirmation with thresholded speech recognition log-likelihood scores, and obtained qualitatively similar results.

**Experiment 3 (Role of Distress Features):** Dialogues often contain timeouts (user silence when system expected response), resets (user asks for current context of dialogue to be abandoned and the system is reinitialized), user requests for help, and other indicators that the dialogue is potentially in trouble. Do such events correlate with low value? We created a state estimator for TOOT that, in addition to our variable I counting informational attributes, counted the number of such distress events in the dialogue. Figure 1(b) presents the optimal value as a function of the number of attributes obtained. Each curve corresponds to a different number of distress features. This figure confirms that the value of the dialogue is lower for states with a higher number of distress features.

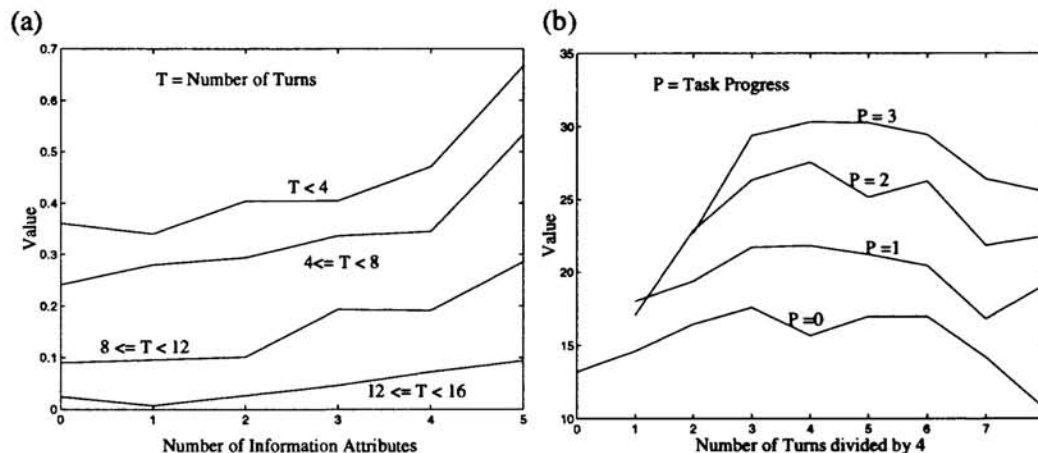

Figure 2: a) Role of Dialogue Length in TOOT. b) Role of Dialogue Length in ELVIS. See description of Experiment 4 in the text for details.

**Experiment 4 (Role of the Dialogue Length):** All other things being equal (e.g. extent of task completion), do users prefer shorter dialogues? To examine this question, we created a state estimator for TOOT that counts the number of informational attributes obtained (variable I as in Experiment 2), and a state estimator for ELVIS that measures "task progress" (a measure analogous to the variable I for TOOT; details omitted). In both cases, a second variable tracks the length of the dialogue.

Figure 2(a) presents the results for TOOT. It plots the optimal value as a function of the number I of informational values; each curve corresponds to a different range of dialogue lengths. It is immediately apparent that the longer the dialogue, the lower the value, and that within the same length of dialogue it is better to have obtained more attributes [4]. Of course, the effect of obtaining more attributes is weak for the longest dialogue length; these are dialogues in which the user is struggling with the system, usually due to multiple speech recognition errors.

Figure 2(b) presents the results for ELVIS from a different perspective. The dialogue length is now the x-axis, while each curve corresponds to a different value of P (task progress). It is immediately apparent that the value increases with task progress. More interestingly, unlike TOOT, there seems to be an "optimal" or appropriate dialogue length for each level of task progress, as seen in the inverse U-shaped curves.

**Experiment 5 (Role of Initiative):** One of the important questions in dialogue theory is how to choose between system and mixed initiative strategies (cf. Section 1). Using our approach on both TOOT and ELVIS data, we were able to confirm previous results [4, 7] showing that system initiative has a higher value than mixed initiative.

**Experiment 6 (Role of Reward Functions):** To test the robustness of our framework, we repeated Experiments 1–4 for TOOT using a new reward function based on the user's perceived task completion. We found that except for a weaker correlation between number of turns and value function, the results were basically the same across the two reward functions.

## 6   Conclusion

This paper presents a new RL-based framework for spoken dialogue systems. Using our framework, we developed RLDS, a general-purpose software tool, and used it for empirical studies on two sets of real dialogues gathered from the TOOT and ELVIS systems. Our results showed that RLDS was able to find sensible policies, that in ELVIS there was an "optimal" length of dialogue, that in TOOT confirmation of attributes was highly correlated with value, that system initiative led to greater user satisfaction than mixed initiative, and that the results were robust to changes in the reward function.

**Acknowledgements:** We give warm thanks to Esther Levin, David McAllester, Roberto Pieraccini, and Rich Sutton for their many contributions to this work.

## Footnotes

[1] However, in recent work we have applied the methodology described here to significantly improve the performance of a new dialogue system [5].

[2]These are not to be confused with the internal states of the spoken dialogue system(s) during the dialogue, which in our view merely contribute observations.

[3]Remember that TOOT can only track its *perceptions* of these attributes, since errors may have occurred in speech recognition.

[4] There is no contradiction with Experiment 2 in this statement, since here we are not separating confirmed and unconfirmed attributes.

## References

[1]  A. W. Biermann and P. M. Long. The composition of messages in speech-graphics interactive systems. In *Proceedings of the 1996 International Symposium on Spoken Dialogue*. 97–100, 1996.

[2]  A. L. Gorin, B. A. Parker, R. M. Sachs and J. G. Wilpon. How May I Help You. In *Proceedings of International Symposium on Spoken Dialogue*. 57–60, 1996.

[3]  E. Levin, R. Pieraccini and W. Eckert. Learning dialogue strategies within the Markov decision process framework. In *Proc. IEEE Workshop on Automatic Speech Recognition and Understanding* 1997.

[4]  D. J. Litman and S. Pan. Empirically Evaluating an Adaptable Spoken Dialogue System. In *Proceedings of the 7th International Conference on User Modeling* 1999.

[5]  S. Singh, M. Kearns, D. Litman, and M. Walker. In preparation.

[6]  R. S. Sutton and A. G. Barto. *Reinforcement Learning: An Introduction* MIT Press, 1998.

[7]  M. A. Walker, J. C. Fromer and S. Narayanan. Learning Optimal Dialogue Strategies: A Case Study of a Spoken Dialogue Agent for Email. In *Proceedings of the 36th Annual Meeting of the Association of Computational Linguistics, COLING/ACL 98* 1345–1352, 1998.

